# A Nonparametric Bayesian Method for Inferring Features From Similarity Judgments

**Daniel J. Navarro**
School of Psychology
University of Adelaide
Adelaide, SA 5005, Australia
daniel.navarro@adelaide.edu.au

**Thomas L. Griffiths**
Department of Psychology
UC Berkeley
Berkeley, CA 94720, USA
tom_griffiths@berkeley.edu

## Abstract

The additive clustering model is widely used to infer the features of a set of stimuli from their similarities, on the assumption that similarity is a weighted linear function of common features. This paper develops a fully Bayesian formulation of the additive clustering model, using methods from nonparametric Bayesian statistics to allow the number of features to vary. We use this to explore several approaches to parameter estimation, showing that the nonparametric Bayesian approach provides a straightforward way to obtain estimates of both the number of features used in producing similarity judgments and their importance.

## 1 Introduction

One of the central problems in cognitive science is determining the mental representations that underlie human inferences. A variety of solutions to this problem are based on the analysis of similarity judgments. By defining a probabilistic model that accounts for the similarity between stimuli based on their representation, statistical methods can be used to infer underlying representations from human similarity judgments. The particular methods used to infer representions from similarity judgments depend on the nature of the underlying representations. For stimuli that are assumed to be represented as points in some psychological space, multidimensional scaling algorithms [1] can be used to translate similarity judgments into stimulus locations. For stimuli that are assumed to be represented in terms of a set of latent features, *additive clustering* is the method of choice.

The original formulation of the additive clustering (ADCLUS) problem [2] is as follows. Assume that we have data in the form of a $n \times n$ similarity matrix $\mathbf{S} = [s_{ij}]$, where $s_{ij}$ is the judged similarity between the $i$th and $j$th of $n$ objects. Similarities are assumed to be symmetric (with $s_{ij} = s_{ji}$) and non-negative, often constrained to lie on the interval $[0, 1]$. These empirical similarities are assumed to be well-approximated by a weighted linear function of common features. Under these assumptions, a representation that uses $m$ features to describe $n$ objects is given by an $n \times m$ matrix $\mathbf{F} = [f_{ik}]$, where $f_{ik} = 1$ if the $i$th object possesses the $k$th feature, and $f_{ik} = 0$ if it is not. Each feature has an associated non-negative saliency weight $\mathbf{w} = (w_1, \ldots, w_m)$. When written in matrix form, the ADCLUS model seeks to uncover a feature matrix $\mathbf{F}$ and a weight vector $\mathbf{w}$ such that $\mathbf{S} \approx \mathbf{FWF}'$, where $\mathbf{W} = \text{diag}(\mathbf{w})$ is a diagonal matrix with nonzero elements corresponding to the saliency weights. In most applications it is assumed that there is a fixed "additive constant", a required feature possessed by all objects.

## 2 A Nonparametric Bayesian ADCLUS Model

To formalize additive clustering as a statistical model, it is standard practice to assume that error terms are *i.i.d.* Gaussian [3], yielding the model:

$$\mathbf{S} = \mathbf{FWF}' + \mathbf{E}, \tag{1}$$

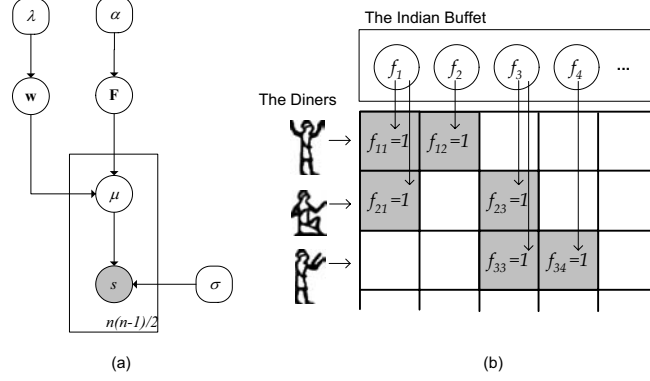

Figure 1: Graphical model representation of the IBP-ADCLUS model. Panel (a) shows the hierarchical structure of the ADCLUS model, and panel (b) illustrates the method by which a feature matrix is generated using the Indian Buffet Process.

where $\mathbf{E} = [\epsilon_{ij}]$ is an $n \times n$ matrix with entries drawn from a Gaussian$(0, \sigma^2)$ distribution. Equation 1 reveals that the additive clustering model is structurally similar to the better-known factor analysis model [4], although there are several differences: most notably the constraints that $\mathbf{F}$ is binary valued, $\mathbf{W}$ is necessarily diagonal and $\mathbf{S}$ is non-negative. In any case, if we define $\mu_{ij} = \sum_k w_k f_{ik} f_{jk}$ to be the similarity predicted by a particular choice of $\mathbf{F}$ and $\mathbf{w}$, then:

$$s_{ij} \,|\, \mathbf{F}, \mathbf{w}, \sigma \quad \sim \quad \text{Normal}(\mu_{ij}, \sigma^2), \tag{2}$$

where $\sigma^2$ is the variance of the Gaussian error distribution. However, self-similarities $s_{ii}$ are not modeled in additive clustering, and are generally fixed to (the same) arbitrary values for both the model and data. It is typical to treat $\sigma^2$ as a fixed parameter [5], and while this could perhaps be improved upon, we leave this open for future research.

In our approach, additive clustering is framed as a form of nonparametric Bayesian inference, in which Equation 2 provides the likelihood function, and the model is completed by placing priors over the weights $\mathbf{w}$ and the feature matrix $\mathbf{F}$. We assume a fixed Gamma prior over feature saliencies though it is straightforward to extend this to other, more flexible, priors. Setting a prior over binary feature matrices $\mathbf{F}$ is more difficult, since there is generally no good reason to assume an upper bound on the number of features that might be relevant to a particular similarity matrix. For this reason we use the "nonparametric" *Indian Buffet Process* (IBP) [6], which provides a proper prior distribution over binary matrices with a fixed number of rows and an unbounded number of columns. The IBP can be understood by imagining an Indian buffet containing an infinite number of dishes. Each customer entering the restaurant samples a number of dishes from the buffet, with a preference for those dishes that other diners have tried. For the $k$th dish sampled by at least one of the first $n - 1$ customers, the probability that the $n$th customer will also try that dish is

$$p(f_{nk} = 1 | \mathbf{F}_{n-1}) = \frac{n_k}{n}, \tag{3}$$

where $\mathbf{F}_{n-1}$ records the choices of the previous customers, and $n_k$ denotes the number of previous customers that have sampled that dish. Being adventurous, the new customer may try some hitherto untasted meals from the infinite buffet on offer. The number of new dishes taken by customer $n$ follows a Poisson$(\alpha/n)$ distribution. The complete IBP-ADCLUS model becomes,

$$
\begin{aligned}
s_{ij} \,|\, \mathbf{F}, \mathbf{w}, \sigma &\quad \sim \quad \text{Normal}(\mu_{ij}, \sigma^2) \\
w_k \,|\, \lambda_1, \lambda_2 &\quad \sim \quad \text{Gamma}(\lambda_1, \lambda_2) \\
\mathbf{F} \,|\, \alpha &\quad \sim \quad \text{IBP}(\alpha).
\end{aligned} \tag{4}
$$

The structure of this model is illustrated graphically in Figure 1(a), and an illustration of the IBP prior is shown in Figure 1(b).

## 3 A Gibbs-Metropolis Sampling Scheme

As a Bayesian formulation of additive clustering, statistical inference in Equation 4 is based on the posterior distribution over feature matrices and saliency vectors, $p(\mathbf{F}, \mathbf{w} \,|\, \mathbf{S})$. Naturally, the ideal

approach is to calculate posterior quantities using exact methods. Unfortunately, this is generally quite difficult, so a natural alternative is to use Markov chain Monte Carlo (MCMC) methods to repeatedly sample from the posterior distribution: estimates of posterior quantities can be made using these samples as proxies for the full distribution. We construct a simple MCMC scheme for the Bayesian ADCLUS model using a combination of Gibbs sampling [7] and more general Metropolis proposals [8].

*Saliency Weights*. We use a Metropolis scheme to resample the saliency weights. If the current saliency is $w_k$, a candidate $w_k^*$ is first generated from a Gaussian$(w_k, 0.05)$ distribution. The value of $w_k$ is then reassigned using the Metropolis update rule. If $\mathbf{w}_{-k}$ denotes the set of all saliencies except $w_k$, this rule is

$$w_k \leftarrow \begin{cases} w_k^* & \text{with probability } a \\ w_k & \text{with probability } 1-a \end{cases}, \quad \text{where } a = \frac{p(\mathbf{S} \mid \mathbf{F}, \mathbf{w}_{-k}, w_k^*) p(w_k^* \mid \lambda)}{p(\mathbf{S} \mid \mathbf{F}, \mathbf{w}_{-k}, w_k) p(w_k \mid \lambda)}. \tag{5}$$

With a Gamma prior, the Metropolis sampler automatically rejects all negative valued $w_k^*$.

*"Pre-Existing" Features*. For features currently possessed by at least one object, assignments are updated using a standard Gibbs sampler: the value of $f_{ik}$ is drawn from the conditional posterior distribution over $f_{ik} \mid \mathbf{S}, \mathbf{F}_{-ik}, \mathbf{w}$. Since feature assignments are discrete, it is easy to find this conditional probability by noting that

$$p(f_{ik} | \mathbf{S}, \mathbf{F}_{-ik}, \mathbf{w}) \propto p(\mathbf{S}|\mathbf{F}, \mathbf{w}) p(f_{ik}|\mathbf{F}_{-ik}), \tag{6}$$

where $\mathbf{F}_{-ik}$ denotes the set of all feature assignments except $f_{ik}$. The first term in this expression is just the likelihood function for the ADCLUS model, and is simple to calculate. Moreover, since feature assignments in the IBP are exchangeable, we can treat the $k$th assignment as if it were the last. Given this, Equation 3 indicates that $p(f_{ik}|\mathbf{F}_{-ik}) = n_{-ik}/n$, where $n_{-ik}$ counts the number of stimuli (besides the $i$th) that currently possess the $k$th feature. The Gibbs sampler deletes all single-stimulus features with probability 1, since $n_{-ik}$ will be zero for one of the stimuli.

*"New" Features*. Since the IBP describes a prior over infinite feature matrices, the resampling procedure needs to accommodate the remaining (infinite) set of features that are not currently represented among the manifest features $\mathbf{F}$. When resampling feature assignments, some finite number of those currently-latent features will become manifest. When sampling from the conditional prior over feature assignments for the $i$th stimulus, we hold the feature assignments fixed for all other stimuli, so this is equivalent to sampling some number of "singleton" features (i.e., features possessed only by stimulus $i$) from the conditional prior, which is Poisson($\alpha/n$) as noted previously.

When working with this algorithm, we typically run several chains. For each chain, we initialize the Gibbs-Metropolis sampler more or less arbitrarily. After a "burn-in" period is allowed for the sampler to converge to a sensible location (i.e., for the state to represent a sample from the posterior), we make a "draw" by recording the state of the sampler, leaving a "lag" of several iterations between successive draws to reduce the autocorrelation between samples. When doing so, it is important to ensure that the Markov chains converge on the target distribution $p(\mathbf{F}, \mathbf{w} \mid \mathbf{S})$. We did so by inspecting the time series plot formed by graphing the log posterior probability of successive samples. To illustrate this, one of the chains used in our simulations (see Section 5) is displayed in Figure 2, with nine parallel chains used for comparison: the time series plot shows no long-term trends, and that different chains are visually indistinguishable from one another. Although elaborations and refinements are possible for both the sampler [9] and the convergence check [10], we have found this approach to be reasonably effective for the moderate-sized problems considered in our applications.

## 4  Four Estimators for the ADCLUS Model

Since the introduction of the additive clustering model, a range of algorithms have been used to infer features, including "subset selection" [2], expectation maximization [3], continuous approximations [11] and stochastic hillclimbing [5] among others. A review, as well as an effective combinatorial search algorithm, is given in [12]. Curiously, while the plethora of *algorithms* available for extracting estimates of $\mathbf{F}$ and $\mathbf{w}$ have been discussed in the literature, the variety in the choice of *estimator* has been largely overlooked, to our knowledge. One advantage of the IBP-ADCLUS approach is that it allows us to discuss a range of different estimators that within a single framework. We will explore estimators based on computing the posterior distribution over $\mathbf{F}$ and $\mathbf{w}$ given $\mathbf{S}$. This includes estimators based on maximum *a posteriori* (MAP) estimation, corresponding to the value of a variable with highest posterior probability, and taking expectations over the posterior distribution.

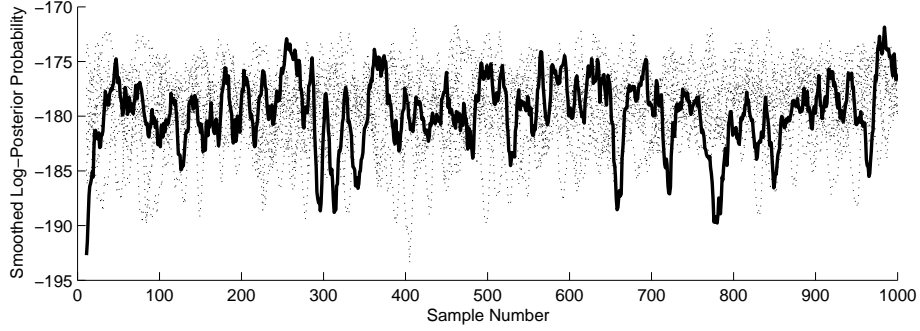

Figure 2: Smoothed time series showing log-posterior probabilities for successive draws from the Gibbs-Metropolis sampler, for simulated similarity data with $n = 16$. The bold line shows a single chain, while the dotted lines show the remaining nine chains.

*Conditional MAP Estimation.* Much of the literature defines an estimator *conditional* on the assumption that the number of features in the model $m$ is fixed [3][11][12]. These approaches seek to estimate the values of $\mathbf{F}$ and $\mathbf{w}$ that jointly maximize some utility function conditional on this known $m$. If we treat the posterior probability to be our measure of utility, the estimators become,

$$\hat{\mathbf{F}}_1, \hat{\mathbf{w}}_1 = \arg \max_{\mathbf{F}, \mathbf{w}} p(\mathbf{F}, \mathbf{w} \mid \mathbf{S}, m) \tag{7}$$

Estimating the dimension is harder. The natural (MAP) estimate for $m$ is easy to state:

$$\hat{m}_1 = \arg \max_m p(m \mid \mathbf{S}) = \arg \max_m \left[ \sum_{\mathbf{F} \in \mathcal{F}_m} \int p(\mathbf{F}, \mathbf{w} \mid \mathbf{S}) \, d\mathbf{w} \right] \tag{8}$$

where $\mathcal{F}_m$ denotes the set of feature matrices containing $m$ unique features. In practice, given the difficulty of working with Equation 8, it is typical to fix $m$ on the basis of intuition, or via some heuristic method.

*MAP Feature Estimation.* In the previous approach, $m$ is given primacy, since $\mathbf{F}$ and $\mathbf{w}$ cannot be estimated until it is known. No distinction is made between $\mathbf{F}$ and $\mathbf{w}$. In many practical situations [13], this does not reflect the priorities of the researcher. Often the feature matrix $\mathbf{F}$ is the psychologically relevant variable, with $\mathbf{w}$ and $m$ being nuisance parameters. In such cases, it is natural to marginalize $\mathbf{w}$ when estimating $\mathbf{F}$, and let the estimated feature matrix itself determine $m$. That is, we first select

$$\hat{\mathbf{F}}_2 = \arg \max_{\mathbf{F}} p(\mathbf{F} \mid \mathbf{S}) = \arg \max_{\mathbf{F}} \left[ \int p(\mathbf{F}, \mathbf{w} \mid \mathbf{S}) d\mathbf{w} \right]. \tag{9}$$

Notice that $\hat{\mathbf{F}}_2$ provides an implicit estimate of $\hat{m}_2$, which may differ from $\hat{m}_1$. The saliencies are estimated after $\hat{\mathbf{F}}_2$ is chosen, via conditional MAP estimation:

$$\hat{\mathbf{w}}_2 = \arg \max_{\mathbf{w}} p(\mathbf{w} \mid \hat{\mathbf{F}}_2, \mathbf{S}). \tag{10}$$

This approach is typical of existing (parametric) Bayesian approaches to additive clustering [5][14], where analytic approximations to $p(\mathbf{F} \mid \mathbf{S})$ are used for expediency.

*Joint MAP Estimation.* Both approaches discussed so far require some aspects of the model to be estimated before others. While the rationales for this constraint differ, both approaches seem sensible. Another approach, not as common in the literature, is to jointly estimate $\mathbf{F}$ and $\mathbf{w}$ without conditioning on $m$, yielding the MAP estimators,

$$\hat{\mathbf{F}}_3, \hat{\mathbf{w}}_3 = \arg \max_{\mathbf{F}, \mathbf{w}} p(\mathbf{F}, \mathbf{w} \mid \mathbf{S}). \tag{11}$$

Early papers [2] recognized that this approach can be prone to overfitting, and thus requires that the prior place some emphasis on parsimony. However, many theoretically-motivated priors (including the IBP) allow the researcher to emphasize parsimony, and some frequentist methods used in ADCLUS-like models apply penalty functions for this reason [15].

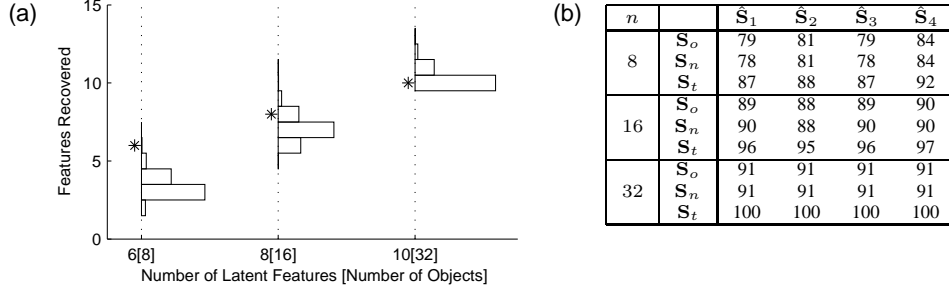

Figure 3: Posterior distributions (a) over the number of features $p(m \mid \mathbf{S}_o)$ in simulations containing $m_t = 6$, $8$ and $10$ features respectively. Variance accounted for (b) by the four similarity estimators $\hat{\mathbf{S}}$, where the target is either the observed training data $\mathbf{S}_o$, a new test data set $\mathbf{S}_n$, or the true similarity matrix $\mathbf{S}_t$.

*Approximate Expectations.* A fourth approach aims to summarize the posterior distribution by looking at the marginal posterior probabilities associated with particular features. The probability that a particular feature $\mathbf{f}_k$ belongs in the representation is given by:

$$p(\mathbf{f}_k \mid \mathbf{S}) = \sum_{\mathbf{F}:\mathbf{f}_k \in \mathbf{F}} p(\mathbf{F} \mid \mathbf{S}). \tag{12}$$

Although this approach has never been applied in the ADCLUS literature, the concept is implicit in more general discussions of mental representation [16] that ask whether or not a specific predicate is likely to be represented. Letting $\hat{r}_k = p(\mathbf{f}_k \mid \mathbf{S})$ denote the posterior probability that feature $\mathbf{f}_k$ is manifest, we can construct a vector $\hat{\mathbf{r}} = [\hat{r}_k]$ that contains these probabilities for all $2^n$ possible features. Although this vector discards the covariation between features across the posterior distribution, it is useful both theoretically (for testing hypotheses about specific features) and pragmatically, since the expected posterior similarities can be written as follows:

$$E[s_{ij}{}^* \mid \mathbf{S}] = \sum_{\mathbf{f}_k} f_{ik} f_{jk} \hat{r}_k \hat{w}_k, \tag{13}$$

where $\hat{w}_k = E\left[w_k \mid \mathbf{f}_k, \mathbf{S}\right]$ denotes the expected saliency for feature $\mathbf{f}_k$ on those occasions when it is represented (Equation 13 relies on the fact that features combine linearly in the ADCLUS model, and is straightforward to derive). In practice, it is impossible to look at all $2^n$ features, so one would typically report only those features for which $\hat{r}_k$ is large. Since these tend to be the features that make the largest contributions to $E[s_{ij}{}^* \mid \mathbf{S}]$, there is a sense in which this approach approximates the expected posterior similarities.

## 5 Recovering Noisy Feature Matrices

By using the IBP-ADCLUS framework, we can compare the performance of the four estimators in a reasonable fashion. Loosely following [12], we generated noisy similarity matrices with $n = 8$, $16$ and $32$ stimuli, based on "true" feature matrices $\mathbf{F}_t$ in which $m_t = 2 \log_2(n)$, where each object possessed each feature with probability $0.5$. Saliency weights $\mathbf{w}_t$ were generated uniformly from the interval $[1, 3]$, but were subsequently rescaled to ensure that the "true" similarities $\mathbf{S}_t$ had variance $1$. Two sets of Gaussian noise were injected into the similarities with fixed $\sigma = 0.3$, ensuring that the noise accounted for approximately 10% of the variance in the "observed" data matrix $\mathbf{S}_o$ and the "new" matrix $\mathbf{S}_n$. We fixed $\alpha = 2$ for all simulations: since the number of manifest features in an IBP model follows a Poisson$(\alpha H_n)$ distribution (where $H_n$ is the $n$th harmonic number) [6], the prior has a strong bias toward parsimony. The prior expected number of features is approximately $5.4$, $6.8$ and $8.1$ (as compared to the true values of $6$, $8$ and $10$).

We approximated the posterior distribution $p(\mathbf{F}, \mathbf{w} \mid \mathbf{S}_1)$, by drawing samples in the following manner. For a given similarity matrix, 10 Gibbs-Metropolis chains were run from different start points, and 1000 samples were drawn from each. The chains were burnt in for 1000 iterations, and a lag of 10 iterations was used between successive samples. Visual inspection suggested that five chains in the $n = 32$ condition did not converge: log-posteriors were low, differed substantially from one

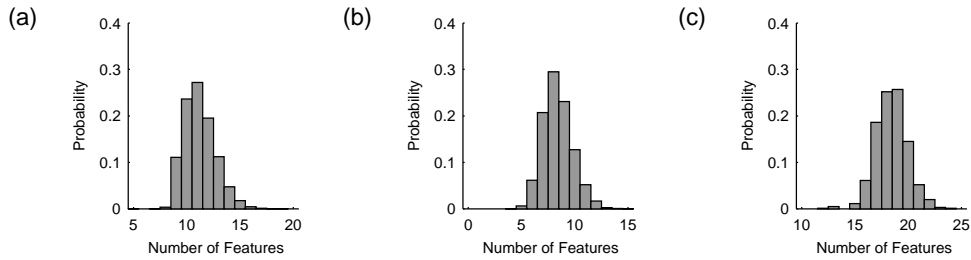

Figure 4: Posterior distributions over the number of features when the Bayesian ADCLUS model is applied to (a) the numbers data, (b) the countries data and (c) the letters data.

Table 1: Two representations of the numbers data. (a) The representation reported in [3], extracted using an EM algorithm with the number of features fixed at eight. (b) The 10 most probable features extracted using the Bayesian ADCLUS model. The first column gives the posterior probability that a particular feature belongs in the representation. The second column displays the average saliency of a feature in the event that it is included.

(a)

| FEATURE | WEIGHT |
|---|---|
| 2   4       8 | 0.444 |
| 0 1 2 | 0.345 |
| 3     6     9 | 0.331 |
| 6 7 8 9 | 0.291 |
| 2 3 4 5 6 | 0.255 |
| 1   3   5   7   9 | 0.216 |
| 1 2 3 4 | 0.214 |
| 4 5 6 7 8 | 0.172 |
| additive constant | 0.148 |

(b)

| FEATURE | PROB. | WEIGHT |
|---|---|---|
| 3     6     9 | 0.79 | 0.326 |
| 2   4       8 | 0.70 | 0.385 |
| 0 1 2 | 0.69 | 0.266 |
| 2 3 4 5 6 | 0.59 | 0.240 |
| 6 7 8 9 | 0.57 | 0.262 |
| 0 1 2 3 4 | 0.42 | 0.173 |
| 2   4   6   8 | 0.41 | 0.387 |
| 1   3   5   7   9 | 0.40 | 0.223 |
| 4 5 6 7 8 | 0.34 | 0.181 |
| 7 8 9 | 0.26 | 0.293 |
| additive constant | 1.00 | 0.075 |

another, and had noticable positive slope. In this case, the estimators were constructed from the five remaining chains.

Figure 3(a) shows the posterior distributions over the number of features $m$ for each of the three simulation conditions. There is a tendency to underestimate the number of features when provided with small similarity matrices, with the modal number being 3, 7 and 10. However, since the posterior estimate of $m$ is below the prior estimate when $n = 8$, it seems this effect is data-driven, as 79% of the variance in the data matrix $\mathbf{S}_o$ can be accounted for using only three features.

Since each approach allows the construction of an estimated similarity matrix $\hat{\mathbf{S}}$, a natural comparison is to look at the proportion of variance this estimate accounts for in the observed data $\mathbf{S}_o$, the novel data set $\mathbf{S}_n$, and the true matrix $\mathbf{S}_t$. In view of the noise model used to construct these matrices, the "ideal" answer for these three should be around 90%, 90% and 100% respectively. When $n = 32$, this profile is observed for all four estimators, suggesting that in this case all four estimators have converged appropriately. For the smaller matrices, the conditional MAP and joint MAP estimators ($\hat{\mathbf{S}}_1$ and $\hat{\mathbf{S}}_3$) agree closely. The MAP feature approach $\hat{\mathbf{S}}_3$ appears to perform slightly better, though the difference is very small. The expectation method $\hat{\mathbf{S}}_4$ provides the best estimate.

## 6    Modeling Empirical Similarities

We now turn to the analysis of empirical data. Since space constraints preclude detailed reporting of all four estimators with respect to all data sets, we limit the discussion to the most novel IBP-ADCLUS estimators, namely the direct estimates of dimensionality provided through Equation 8, and the features extracted via "approximate expectation".

*Featural representations of numbers.* A standard data set used in evaluating additive clustering models measures the conceptual similarity of the numbers 0 through 9 [17]. This data set is often used as a benchmark due to the complex interrelationships between the numbers. Table 1(a) shows an eight-feature representation of these data, taken from [3] who applied a maximum likelihood approach. This representation explains 90.9% of the variance, with features corresponding to arith-

Table 2: Featural representation of the similarity between 16 countries. The table shows the eight highest-probability features extracted by the Bayesian ADCLUS model. Each column corresponds to a single feature, with the associated probabilities and saliencies shown below. The average weight associated with the additive constant is 0.035.

| | FEATURE | | | | | | | |
|---|---|---|---|---|---|---|---|---|
| | Italy Germany Spain | Vietnam China Japan Philippines Indonesia | Germany Russia USA China Japan | Zimbabwe Nigeria | Zimbabwe Nigeria Cuba Jamaica Iraq Libya | Iraq Libya | Zimbabwe Nigeria Iraq Libya | Philippines Indonesia |
| PROB. | 1.00 | 1.00 | 0.99 | 0.62 | 0.52 | 0.36 | 0.33 | 0.25 |
| WEIGHT | 0.593 | 0.421 | 0.267 | 0.467 | 0.209 | 0.373 | 0.299 | 0.311 |

Table 3: Featural representation of the perceptual similarity between 26 capital letters. The table shows the ten highest-probability features extracted by the Bayesian ADCLUS model. Each column corresponds to a single feature, with the associated probabilities and saliencies shown below. The average weight associated with the additive constant is 0.003.

| | FEATURE | | | | | | | | | |
|---|---|---|---|---|---|---|---|---|---|---|
| | M N W | I L T | C G | D O Q | P R | E F | E H | K X | B G R | C J U |
| PROB. | 1.00 | 0.99 | 0.99 | 0.99 | 0.99 | 0.99 | 0.99 | 0.99 | 0.98 | 0.92 |
| WEIGHT | 0.686 | 0.341 | 0.623 | 0.321 | 0.465 | 0.653 | 0.322 | 0.427 | 0.226 | 0.225 |

metic concepts and to numerical magnitude. Fixing $\sigma = 0.05$, and $\alpha = 0.5$, we drew 10,000 lagged samples to construct estimates. Although the posterior probability is spread over a large number of feature matrices, 92.6% of sampled matrices had between 9 and 13 features. The modal number of represented features was $\hat{m}_1 = 11$, with 27.2% of the posterior mass. The posterior distribution over the number of features is shown in Figure 4(a). Since none of the existing literature has used the "approximate expectation" approach to find highly probable features, it is useful to note the strong similarities between Table 1(a) and Table 1(b), which reports the ten highest-probability features across the entire posterior distribution. Applying this approach to obtain an estimate of the posterior predictive similarities $\hat{S}_4$ revealed that this matrix accounts for 97.4% of the variance in the data.

*Featural representations of countries.* A second application is to human forced-choice judgments of the similarities between 16 countries [18]. In this task, participants were shown lists of four countries and asked to pick out the two countries most similar to each other. Applying the Bayesian model to these data with $\sigma = 0.1$ reveals that only eight features appear in the representation more than 25% of the time. Given this, it is not surprising that the posterior distribution over the number of features, shown in Figure 4 (b), indicates that the modal number of features is eight. The eight most probable features are listed in Table 2. The "approximate expectation" method explains 85.4% of the variance, as compared to the 78.1% found by a MAP feature approach [18]. The features are interpretable, corresponding to a range of geographical, historical, and economic regularities.

*Featural representations of letters.* As a third example, we analyzed a somewhat larger data set, consisting of kindergarten children's assessment of the perceptual similarity of the 26 capital letters [19]. In this case, we used $\sigma = 0.05$, and the Bayesian model accounted for 89.2% of the variance in the children's similarity judgments. The posterior distribution over the number of represented features is shown in Figure 4(c). Table 3 shows the ten features that appeared in more than 90% of samples from the posterior. The model recovers an extremely intuitive set of overlapping features. For example, it picks out the long strokes in I, L, and T, and the elliptical forms of D, O, and Q.

## 7 Discussion

Learning how similarity relations are represented is a difficult modeling problem. Additive clustering provides a framework for learning featural representations of stimulus similarity, but remains underused due to the difficulties associated with the inference. By adopting a Bayesian approach

to additive clustering, we are able to obtain a richer characterization of the structure behind human similarity judgments. Moreover, by using nonparametric Bayesian techniques to place a prior distribution over infinite binary feature matrices via the Indian Buffet Process, we can allow the data to determine the number of features that the algorithm recovers. This is theoretically important as well as pragmatically useful. As noted by [16], people are capable of recognizing that individual stimuli possess an arbitrarily large number of characteristics, but in any particular context will make judgments using only a finite, usually small number of properties that form part of our current mental representation. In other words, by moving to a Bayesian nonparametric form, we are able to bring the ADCLUS model closer to the kinds of assumptions that are made by psychological theories.

**Acknowledgements.** TLG was supported by NSF grant number 0631518, and DJN by ARC grants DP-0451793 and DP-0773794. We thank Nancy Briggs, Simon Dennis and Michael Lee for helpful comments on this work.

# References

[1] W. S. Torgerson. *Theory and Methods of Scaling*. Wiley, New York, 1958.

[2] R. N. Shepard and P. Arabie. Additive clustering: Representation of similarities as combinations of discrete overlapping properties. *Psychological Review*, 86:87–123, 1979.

[3] J. B. Tenenbaum. Learning the structure of similarity. In D. S. Touretzky, M. C. Mozer, and M. E. Hasselmo, editors, *Advances in Neural Information Processing Systems*, volume 8, pages 3–9. MIT Press, Cambridge, MA, 1996.

[4] L. L. Thurstone. *Multiple-Factor Analysis*. University of Chicago Press, Chicago, 1947.

[5] M. D. Lee. Generating additive clustering models with limited stochastic complexity. *Journal of Classification*, 19:69–85, 2002.

[6] T. L. Griffiths and Z. Ghahramani. Infinite latent feature models and the Indian buffet process. Technical Report 2005-001, Gatsby Computational Neuroscience Unit, 2005.

[7] S. Geman and D. Geman. Stochastic relaxation, Gibbs distributions, and the Bayesian restoration of images. *IEEE Transactions on Pattern Analysis and Machine Intelligence*, 6:721–741, 1984.

[8] N. Metropolis, A. W. Rosenbluth, M. N. Rosenbluth, A. H. Teller, and E. Teller. Equations of state calculations by fast computing machines. *Journal of Chemical Physics*, 21:1087–1092, 1953.

[9] Q.-M. Shao M.-H. Chen and J. G. Ibrahim. *Monte Carlo Methods in Bayesian Computation*. Springer, New York, 2000.

[10] M. K. Cowles and B. P. Carlin. Markov chain Monte Carlo convergence diagnostics: A comparative review. *Journal of the American Statistical Association*, 91:833–904, 1996.

[11] P. Arabie and J. Douglas Carroll. MAPCLUS: A mathematical programming approach to fitting the ADCLUS model. *Psychometrika*, 45:211–235, 1980.

[12] W. Ruml. Constructing distributed representations using additive clustering. In *Advances in Neural Information Processing Systems 14*, Cambridge, MA, 2001. MIT Press.

[13] M. D. Lee and D. J. Navarro. Extending the ALCOVE model of category learning to featural stimulus domains. *Psychonomic Bulletin and Review*, 9:43–58, 2002.

[14] D. J. Navarro. *Representing Stimulus Similarity*. Ph.D. Thesis, University of Adelaide, 2003.

[15] L. E. Frank and W. J. Heiser. Feature selection in Feature Network Models: Finding predictive subsets of features with the Positive Lasso. *British Journal of Mathematical and Statistical Psychology*, in press.

[16] D. L. Medin and A. Ortony. Psychological essentialism. In *Similarity and Analogical Reasoning*. Cambridge University Press, New York, 1989.

[17] R. N. Shepard, D. W. Kilpatric, and J. P. Cunningham. The internal representation of numbers. *Cognitive Psychology*, 7:82–138, 1975.

[18] D. J. Navarro and M. D. Lee. Commonalities and distinctions in featural stimulus representations. In *Proceedings of the 24th Annual Conference of the Cognitive Science Society*, pages 685–690, Mahwah, NJ, 2002. Lawrence Erlbaum.

[19] E. Z. Rothkopf. A measure of stimulus similarity and errors in some paired-associate learning tasks. *Journal of Experimental Psychology*, 53:94–101, 1957.
